# A Support Vector Method for Clustering

**Asa Ben-Hur**
Faculty of IE and Management
Technion, Haifa 32000, Israel

**David Horn**
School of Physics and Astronomy
Tel Aviv University, Tel Aviv 69978, Israel

**Hava T. Siegelmann**
Lab for Inf. & Decision Systems
MIT Cambridge, MA 02139, USA

**Vladimir Vapnik**
AT&T Labs Research
100 Schultz Dr., Red Bank, NJ 07701, USA

## Abstract

We present a novel method for clustering using the support vector machine approach. Data points are mapped to a high dimensional feature space, where support vectors are used to define a sphere enclosing them. The boundary of the sphere forms in data space a set of closed contours containing the data. Data points enclosed by each contour are defined as a cluster. As the width parameter of the Gaussian kernel is decreased, these contours fit the data more tightly and splitting of contours occurs. The algorithm works by separating clusters according to valleys in the underlying probability distribution, and thus clusters can take on arbitrary geometrical shapes. As in other SV algorithms, outliers can be dealt with by introducing a soft margin constant leading to smoother cluster boundaries. The structure of the data is explored by varying the two parameters. We investigate the dependence of our method on these parameters and apply it to several data sets.

## 1 Introduction

Clustering is an ill-defined problem for which there exist numerous methods [1, 2]. These can be based on parametric models or can be non-parametric. Parametric algorithms are usually limited in their expressive power, i.e. a certain cluster structure is assumed. In this paper we propose a non-parametric clustering algorithm based on the support vector approach [3], which is usually employed for supervised learning. In the papers [4, 5] an SV algorithm for characterizing the support of a high dimensional distribution was proposed. As a by-product of the algorithm one can compute a set of contours which enclose the data points. These contours were interpreted by us as cluster boundaries [6]. In [6] the number of clusters was predefined, and the value of the kernel parameter was not determined as part of the algorithm. In this paper we address these issues. The first stage of our Support Vector Clustering (SVC) algorithm consists of computing the sphere with minimal radius which encloses the data points when mapped to a high dimensional feature space. This sphere corresponds to a set of contours which enclose the points in input space. As the width parameter of the Gaussian kernel function that represents the map to feature space

is decreased, this contour breaks into an increasing number of disconnected pieces. The points enclosed by each separate piece are interpreted as belonging to the same cluster. Since the contours characterize the support of the data, our algorithm identifies valleys in its probability distribution. When we deal with overlapping clusters we have to employ a soft margin constant, allowing for "outliers". In this parameter range our algorithm is similar to the space clustering method [7]. The latter is based on a Parzen window estimate of the probability density, using a Gaussian kernel and identifying cluster centers with peaks of the estimator.

## 2 Describing Cluster Boundaries with Support Vectors

In this section we describe an algorithm for representing the support of a probability distribution by a finite data set using the formalism of support vectors [5, 4]. It forms the basis of our clustering algorithm. Let $\{\mathbf{x}_i\} \subseteq \chi$ be a data-set of $N$ points, with $\chi \subseteq \mathbb{R}^d$, the input space. Using a nonlinear transformation $\Phi$ from $\chi$ to some high dimensional feature-space, we look for the smallest enclosing sphere of radius $R$, described by the constraints: $||\Phi(\mathbf{x}_i) - \mathbf{a}||^2 \leq R^2 \ \forall i$ , where $|| \cdot ||$ is the Euclidean norm and $\mathbf{a}$ is the center of the sphere. Soft constraints are incorporated by adding slack variables $\xi_j$:

$$||\Phi(\mathbf{x}_j) - \mathbf{a}||^2 \leq R^2 + \xi_j \tag{1}$$

with $\xi_j \geq 0$. To solve this problem we introduce the Lagrangian

$$L = R^2 - \sum_j (R^2 + \xi_j - ||\Phi(\mathbf{x}_j) - \mathbf{a}||^2)\beta_j - \sum \xi_j \mu_j + C \sum \xi_j \, , \tag{2}$$

where $\beta_j \geq 0$ and $\mu_j \geq 0$ are Lagrange multipliers, $C$ is a constant, and $C \sum \xi_j$ is a penalty term. Setting to zero the derivative of $L$ with respect to $R$, $\mathbf{a}$ and $\xi_j$, respectively, leads to

$$\sum_j \beta_j = 1 \, , \tag{3}$$

$$\mathbf{a} = \sum_j \beta_j \Phi(\mathbf{x}_j) \, , \tag{4}$$

$$\beta_j = C - \mu_j \tag{5}$$

The KKT complementarity conditions [8] result in

$$\xi_j \mu_j = 0 \tag{6}$$

$$(R^2 + \xi_j - ||\Phi(\mathbf{x}_j) - \mathbf{a}||^2)\beta_j = 0 \tag{7}$$

A point $\mathbf{x}_i$ with $\xi_i > 0$ is outside the feature-space sphere (cf. equation 1). Equation (6) states that such points $\mathbf{x_i}$ have $\mu_i = 0$, so from equation (5) $\beta_i = C$. A point with $\xi_i = 0$ is inside or on the surface of the feature space sphere. If its $\beta_i \neq 0$ then equation 7 implies that the point $\mathbf{x}_i$ is on the surface of the feature space sphere. In this paper any point with $0 < \beta_i < C$ will be referred to as a *support vector* or SV; points with $\beta_i = C$ will be called *bounded support vectors* or bounded SVs. This is to emphasize the role of the support vectors as delineating the boundary. Note that when $C \geq 1$ no bounded SVs exist because of the constraint $\sum \beta_i = 1$.

Using these relations we may eliminate the variables $R$, $\mathbf{a}$ and $\mu_j$, turning the Lagrangian into the Wolfe dual which is a function of the variables $\beta_j$:

$$W = \sum_j \Phi(\mathbf{x}_j)^2 \beta_j - \sum_{i,j} \beta_i \beta_j \Phi(\mathbf{x}_i) \cdot \Phi(\mathbf{x}_j) \tag{8}$$

Since the variables $\mu_j$ don't appear in the Lagrangian they may be replaced with the constraints:

$$0 \le \beta_j \le C. \tag{9}$$

We follow the SV method and represent the dot products $\Phi(\mathbf{x}_i) \cdot \Phi(\mathbf{x}_j)$ by an appropriate Mercer kernel $K(\mathbf{x}_i, \mathbf{x}_j)$. Throughout this paper we use the Gaussian kernel

$$K(\mathbf{x}_i, \mathbf{x}_j) = e^{-q\|\mathbf{x}_i - \mathbf{x}_j\|^2}, \tag{10}$$

with width parameter $q$. As noted in [5], polynomial kernels do not yield tight contour representations of a cluster. The Lagrangian $W$ is now written as:

$$W = \sum_j K(\mathbf{x}_j, \mathbf{x}_j)\beta_j - \sum_{i,j} \beta_i \beta_j K(\mathbf{x}_i, \mathbf{x}_j). \tag{11}$$

At each point $\mathbf{x}$ we define its distance, when mapped to feature space, from the center of the sphere:

$$R^2(\mathbf{x}) = \|\Phi(\mathbf{x}) - \mathbf{a}\|^2. \tag{12}$$

In view of (4) and the definition of the kernel we have:

$$R^2(\mathbf{x}) = K(\mathbf{x}, \mathbf{x}) - 2\sum_j \beta_j K(\mathbf{x}_j, \mathbf{x}) + \sum_{i,j} \beta_i \beta_j K(\mathbf{x}_i, \mathbf{x}_j). \tag{13}$$

The radius of the sphere is:

$$R = \{R(\mathbf{x}_i) \mid \mathbf{x}_i \text{ is a support vector}\}. \tag{14}$$

In practice, one takes the average over all support vectors. The contour that encloses the cluster in data space is the set

$$\{\mathbf{x} \mid R(\mathbf{x}) = R\}. \tag{15}$$

A data point $\mathbf{x_i}$ is a bounded SV if $R(\mathbf{x_i}) > R$. Note that since we use a Gaussian kernel for which $K(\mathbf{x}, \mathbf{x}) = 1$, our feature space is a unit sphere; thus its intersection with a sphere of radius $R < 1$ can also be defined as an intersection by a hyperplane, as in conventional SVM.

The shape of the enclosing contours in input space is governed by two parameters, $q$ and $C$. Figure 1 demonstrates that, as $q$ is increased, the enclosing contours form tighter fits to the data. Figure 2 describes a situation that necessitated introduction of outliers, or bounded SV, by allowing for $C < 1$. As $C$ is decreased not only does the number of bounded SVs increase, but their influence on the shape of the cluster contour decreases (see also [6]). The number of support vectors depends on both $q$ and $C$. For fixed $q$, as $C$ is decreased, the number of SVs decreases since some of them turn into bounded SVs and the resulting shapes of the contours become smoother.

We denote by $n_{sv}, n_{bsv}$ the number of support vectors and bounded support vectors, respectively, and note the following result:

**Proposition 2.1** [4]

$$n_{bsv} + n_{sv} \ge 1/C, \quad n_{bsv} < 1/C \tag{16}$$

This is an immediate consequence of the constraints (3) and (9). In fact, we have found empirically that

$$n_{bsv}(q, C) = \max(0, 1/C - n_0), \tag{17}$$

where $n_0 > 0$ may be a function of $q$ and $N$. This was observed for artificial and real data sets. Moreover, we have also observed that

$$n_{sv} = a/C + b, \tag{18}$$

where $a$ and $b$ are functions of $q$ and $N$. The linear behavior of $n_{bsv}$ continues until $n_{bsv} + n_{sv} = N$.

# 3 Support Vector Clustering (SVC)

In this section we go through a set of examples demonstrating the use of SVC. We begin with a data set in which the separation into clusters can be achieved without outliers, i.e. $C = 1$. As seen in Figure 1, as $q$ is increased the shape of the boundary curves in data-space varies. At several $q$ values the enclosing contour splits, forming an increasing number of connected components. We regard each component as representing a single cluster. While in this example clustering looks hierarchical, this is not strictly true in general.

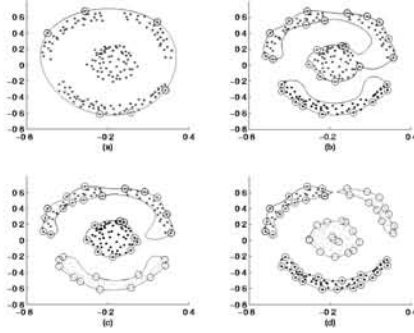

Figure 1: Data set contains 183 points. A Gaussian kernel was used with $C = 1.0$. SVs are surrounded by small circles. (a): $q = 1$ (b): $q = 20$ (c): $q = 24$ (d): $q = 48$.

In order to label data points into clusters we need to identify the connected components. We define an adjacency matrix $A_{ij}$ between pairs of points $\mathbf{x}_i$ and $\mathbf{x}_j$:

$$A_{ij} = \begin{cases} 1 & \text{if for all } \mathbf{y} \text{ on the line segment connecting } \mathbf{x}_i \text{and } \mathbf{x}_j \; R(\mathbf{y}) \leq R \\ 0 & \text{otherwise.} \end{cases} \tag{19}$$

Clusters are then defined as the connected components of the graph induced by $A$. This labeling procedure is justified by the observation that nearest neighbors in data space can be connected by a line segment that is contained in the high dimensional sphere. Checking the line segment is implemented by sampling a number of points on the segment (a value of 10 was used in the numerical experiments). Note that bounded SVs are not classified by this procedure; they can be left unlabeled, or classified e.g., according to the cluster to which they are closest to. We adopt the latter approach.

The cluster description algorithm provides an estimate of the support of the underlying probability distribution [4]. Thus we distinguish between clusters according to gaps in the support of the underlying probability distribution. As $q$ is increased the support is characterized by more detailed features, enabling the detection of smaller gaps. Too high a value of $q$ may lead to overfitting (see figure 2(a)), which can be handled by allowing for bounded SVs (figure 2(b)): letting some of the data points be bounded SVs creates smoother contours, and facilitates contour splitting at low values of $q$.

## 3.1 Overlapping clusters

In many data sets clusters are strongly overlapping, and clear separating valleys as in Figures 1 and 2 are not present. Our algorithm is useful in such cases as well, but a slightly different interpretation is required. First we note that equation (15) for the enclosing contour can be expressed as $\{\mathbf{x} \mid \sum_i \beta_i K(\mathbf{x}_i, \mathbf{x}) = \rho\}$, where $\rho$ is determined by the value of this sum on the support vectors. The set of points enclosed by the contour is:

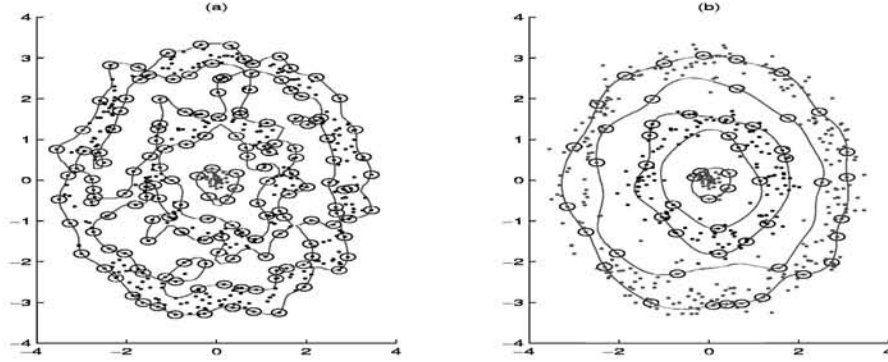

Figure 2: Clustering with and without outliers. The inner cluster is composed of 50 points generated by a Gaussian distribution. The two concentric rings contain 150/300 points, generated by a uniform angular distribution and radial Gaussian distribution. (a) The rings cannot be distinguished when $C = 1$. Shown here is $q = 3.5$, the lowest $q$ value that leads to separation of the inner cluster. (b) Outliers allow easy clustering. The parameters are $1/(NC) = 0.3$ and $q = 1.0$. SVs are surrounded by small ellipses.

$\{\mathbf{x} \mid \sum_i \beta_i K(\mathbf{x}_i, \mathbf{x}) > \rho\}$. In the extreme case when almost all data points are bounded SVs, the sum in this expression is approximately

$$p(\mathbf{x}) = \frac{1}{N} \sum_i K(\mathbf{x}_i, \mathbf{x}). \tag{20}$$

This is recognized as a Parzen window estimate of the density function (up to a normalization factor, if the kernel is not appropriately normalized). The contour will then enclose a small number of points which correspond to the maximum of the Parzen-estimated density. Thus in the high bounded SVs regime we find a dense *core* of the probability distribution.

In this regime our algorithm is closely related to an algorithm proposed by Roberts [7]. He defines cluster centers as maxima of the Parzen window estimate $p(\mathbf{x})$. He shows that in his approach, which goes by the name of scale-space clustering, as $q$ is increased the number of maxima increases. The Gaussian kernel plays an important role in his analysis: it is the only kernel for which the number of maxima (hence the number of clusters) is a monotonically non-decreasing function of $q$ (see [7] and references therein).

The advantage of SVC over Roberts' method is that we find a region, rather than just a peak, and that instead of solving a problem with many local maxima, we identify the core regions by an SV method with a global optimal solution. We have found examples where a local maximum is hard to identify by Roberts' method.

### 3.2 The iris data

We ran SVC on the iris data set [9], which is a standard benchmark in the pattern recognition literature. It can be obtained from the UCI repository [10]. The data set contains 150 instances, each containing four measurements of an iris flower. There are three types of flowers, represented by 50 instances each. We clustered the data in a two dimensional subspace formed by the first two principal components. One of the clusters is linearly separable from the other two at $q = 0.5$ with no bounded SVs. The remaining two clusters have significant overlap, and were separated at $q = 4.2, 1/(NC) = 0.55$, with 4 misclassifications. Clustering results for an increasing number of principal components are reported

Table 1: Performance of SVC on the iris data for a varying number of principal components.

| Principal components | $q$ | $1/(NC)$ | SVs | bounded SVs | misclassified |
|---|---|---|---|---|---|
| 1-2 | 4.2 | 0.55 | 20 | 72 | 4 |
| 1-3 | 7.0 | 0.70 | 23 | 94 | 4 |
| 1-4 | 9.0 | 0.75 | 34 | 96 | 14 |

in Table 1. Note that as the number of principal components is increased from 3 to 4 there is a degradation in the performance of the algorithm - the number of misclassifications increases from 4 to 14. Also note the increase in the number of support vectors and bounded support vectors required to obtain contour splitting. As the dimensionality of the data increases a larger number of support vectors is required to describe the contours. Thus if the data is sparse, it is better to use SVC on a low dimensional representation, obtained, e.g. by principal component analysis [2]. For comparison we quote results obtained by other non-parametric clustering algorithms: the information theoretic approach of [11] leads to 5 miscalssification and the SPC algorithm of [12] has 15 misclassifications.

## 4 Varying $q$ and $C$

SVC was described for fixed values of $q$ and $C$, and a method for exploring parameter space is required. We can work with SVC in an agglomerative fashion, starting from a large value of $q$, where each point is in a different cluster, and decreasing $q$ until there is a single cluster. Alternatively we may use the divisive approach, by starting from a small value of $q$ and increasing it. The latter seems more efficient since meaningful clustering solutions (see below for a definition of this concept), usually have a small number of clusters.

The following is a qualitative schedule for varying the parameters. One may start with a small value of $q$ where only one cluster occurs: $q = 1/\max_{i,j} \|\mathbf{x}_i - \mathbf{x}_j\|^2$. $q$ is then increased to look for values at which a cluster contour splits. When single point clusters start to break off or a large number of support vectors is obtained (overfitting, as in Figure 2(a)) $1/C$ is increased.

An important issue in the divisive approach is the decision when to stop dividing the clusters. An algorithm for this is described in [13]. After clustering the data they partition the data into two sets with some sizable overlap, perform clustering on these smaller data sets and compute the average overlap between the two clustering solutions for a number of partitions. Such validation can be performed here as well. However, we believe that in our SV setting it is natural to use the number of support vectors as an indication of a meaningful solution, since their (small) number is an indication of good generalization. Therefore we should stop the algorithm when the fraction of SVs exceeds some threshold. If the clustering solution is stable with respect to changes in the parameters this is also an indication of meaningful clustering.

The quadratic programming problem of equation (2) can be solved by the SMO algorithm [14] which was recently proposed as an efficient tool for solving such problems in SVM training. Some minor modifications are required to adapt it to the problem that we solve here [4]. Benchmarks reported in [14] show that this algorithm converges in most cases in $O(N^2)$ kernel evaluations. The complexity of the labeling part of the algorithm is $O(N^2 d)$, so that the overall complexity is $O(N^2 d)$. We also note that the memory requirements of the SMO algorithm are low - it can be implemented using $O(1)$ memory at the cost of a decrease in efficiency, which makes our algorithm useful even for very large data-sets.

## 5 Summary

The SVC algorithm finds clustering solutions together with curves representing their boundaries via a description of the support or high density regions of the data. As such, it separates between clusters according to gaps or low density regions in the probability distribution of the data, and makes no assumptions on cluster shapes in input space.

SVC has several other attractive features: the quadratic programming problem of the cluster description algorithm is convex and has a globally optimal solution, and, like other SV algorithms, SVC can deal with noise or outliers by a margin parameter, making it robust with respect to noise in the data.

## References

[1] A.K. Jain and R.C. Dubes. *Algorithms for clustering data*. Prentice Hall, Englewood Cliffs, NJ, 1988.

[2] K. Fukunaga. *Introduction to Statistical Pattern Recognition*. Academic Press, San Diego, CA, 1990.

[3] V. Vapnik. *The Nature of Statistical Learning Theory*. Springer, N.Y., 1995.

[4] B. Schölkopf, R.C. Williamson, A.J. Smola, and J. Shawe-Taylor. SV estimation of a distribution's support. In *Neural Information Processing Systems*, 2000.

[5] D.M.J. Tax and R.P.W. Duin. Support vector domain description. *Pattern Recognition Letters*, 20:1991–1999, 1999.

[6] A. Ben-Hur, D. Horn, H.T. Siegelmann, and V. Vapnik. A support vector clustering method. In *International Conference on Pattern Recognition*, 2000.

[7] S.J. Roberts. Non-parametric unsupervised cluster analysis. *Pattern Recognition*, 30(2):261–272, 1997.

[8] R. Fletcher. *Practical Methods of Optimization*. Wiley-Interscience, Chichester, 1987.

[9] R.A. Fisher. The use of multiple measurements in taxonomic problems. *Annual Eugenics*, 7:179–188, 1936.

[10] C.L. Blake and C.J. Merz. UCI repository of machine learning databases, 1998.

[11] N. Tishby and N. Slonim. Data clustering by Markovian relaxation and the information bottleneck method. In *Neural Information Processing Systems*, 2000.

[12] M. Blatt, S. Wiseman, and E. Domany. Data clustering using a model granular magnet. *Neural Computation*, 9:1804–1842, 1997.

[13] S. Dubnov, R. El-Yaniv, Y. Gdalyahu, E. Schneidman, N. Tishby, and G. Yona. A new nonparametric pairwise clustering algorithm. Submitted to *Machine Learning*.

[14] J. Platt. Fast training of support vector machines using sequential minimal optimization. In B. Schölkopf, C. J. C. Burges, and A. J. Smola, editors, *Advances in Kernel Methods — Support Vector Learning*, pages 185–208, Cambridge, MA, 1999. MIT Press.
